# Towards Property-Based Classification of Clustering Paradigms

**Margareta Ackerman, Shai Ben-David, and David Loker**
D.R.C. School of Computer Science
University of Waterloo, Canada
{mackerma, shai, dloker}@cs.uwaterloo.ca

## Abstract

Clustering is a basic data mining task with a wide variety of applications. Not surprisingly, there exist many clustering algorithms. However, clustering is an ill defined problem - given a data set, it is not clear what a "correct" clustering for that set is. Indeed, different algorithms may yield dramatically different outputs for the same input sets. Faced with a concrete clustering task, a user needs to choose an appropriate clustering algorithm. Currently, such decisions are often made in a very ad hoc, if not completely random, manner. Given the crucial effect of the choice of a clustering algorithm on the resulting clustering, this state of affairs is truly regrettable. In this paper we address the major research challenge of developing tools for helping users make more informed decisions when they come to pick a clustering tool for their data. This is, of course, a very ambitious endeavor, and in this paper, we make some first steps towards this goal. We propose to address this problem by distilling abstract properties of the input-output behavior of different clustering paradigms.

In this paper, we demonstrate how abstract, intuitive properties of clustering functions can be used to taxonomize a set of popular clustering algorithmic paradigms. On top of addressing deterministic clustering algorithms, we also propose similar properties for randomized algorithms and use them to highlight functional differences between different common implementations of $k$-means clustering. We also study relationships between the properties, independent of any particular algorithm. In particular, we strengthen Kleinberg's famous impossibility result, while providing a simpler proof.

## 1 Introduction

In spite of the wide use of clustering in many practical applications, currently, there exists no principled method to guide the selection of a clustering algorithm. Of course, users are aware of the costs involved in employing different clustering algorithms (software purchasing costs, running times, memory requirements, needs for data preprocessing etc.) but there is very little understanding of the differences in the *outcomes* that these algorithms may produce. We focus on that aspect - the input-output properties of different clustering algorithms.

The choice of an appropriate clustering should, of course, be task dependent. A clustering that works well for one task may be unsuitable for another. Even more than for supervised learning, for clustering, the choice of an algorithm must incorporate domain knowledge. While some domain knowledge is embedded in the choice of similarity between domain elements (or the embedding of these elements into some Euclidean space), there is still a large variance in the behavior of difference clustering paradigms over a fixed similarity measure.

For some clustering tasks, there is a natural clustering objective function that one may wish to optimize (like $k$-means for vector quantization coding tasks), but very often the task does not readily translate into a corresponding objective function. Often users are merely searching for a meaningful clustering, without a prior preference for any specific objective function. Many (if not most) common clustering paradigms do not optimize any clearly defined objective utility, either because no such objective is defined (like in the case of, say, single linkage clustering) or because optimizing the most relevant objective is computationally infeasible. To overcome computation infeasibility, the algorithms end up carrying out a heuristic whose outcome may be quite different than the actual objective-based optimum (that is the case with the k-means algorithm as well as with spectral clustering algorithms). What seems to be missing is a clear understanding of the differences in clustering outputs in terms of intuitive and usable properties.

We propose a different approach to providing guidance to clustering users by identifying significant properties of clustering functions that, on one hand distinguish between different clustering paradigms, and on the other hand are intended to be relevant to the domain knowledge that a user might have access to. Based on domain expertise users could then choose which properties they want an algorithm to satisfy, and determine which algorithms meet their requirements.

Our vision is that ultimately, there would be a sufficiently rich set of properties that would provide a detailed, property-based, taxonomy of clustering methods, that could, in turn, be used as guidelines for a wide variety of clustering applications. This is a very ambitious enterprize, but that should not deter researchers from addressing it. This paper takes a step towards that goal by using natural properties to examine some popular clustering approaches.

We present a taxonomy for common deterministic clustering functions with respect to the properties that we propose. We also show how to extend this framework to the randomized clustering algorithms, and use these properties to distinguish between two $k$-means heuristics.

We also study relationships between the properties, independent of any particular algorithm. In particular, we strengthen Kleinberg's impossibility result[8] using a relaxation of one of the properties that he proposed.

## 1.1 Previous work

Our work follows a theoretical study of clustering that began with Kleinberg's impossibility result [8], in which he proposes three candidate axioms of clustering and shows that no clustering function can simultaneously satisfy these three axioms. Ackerman and Ben-David [1] subsequently showed these axioms to be consistent in the setting of clustering quality measures. [1] also proposes to make a distinction between clustering "axioms" and clustering "properties", where the axioms are the features that define which partitionings are worthy of the name "clustering", and the properties vary between different clustering paradigms and may be used to construct a taxonomy of clustering algorithms. We adopt that approach here.

There are previous results that provide some property based characterizations of a specific clustering algorithm. In 1975, Jardine and Sibson [6] gave a characterization of single linkage. Last year, Bosagh Zadeh and Ben-David [3] characterize single-linkage within Kleinberg's framework of clustering functions using a special invariance property ("path distance coherence"). Very recently, Ackerman, Ben-David and Loker provided a characterization of the family of linkage-based clustering in terms of a few natural properties [2].

Some heuristics have been proposed as a means of distinguishing between the output of clustering algorithms on specific data. These approaches require running the algorithms, and then selecting an algorithm based on the outputs that they produce. In particular, validity criteria can be used to evaluate the output of clustering algorithms. These measures can be used to select a clustering algorithm by choosing the one that yields the highest quality clustering [10]. However, the result only applies to the original data, and there are no guarantees on the quality of the output of these algorithms on any other data.

## 2 Definitions and Formal Framework

Clustering is wide and heterogenous domain. For most of this paper, we focus on a basic sub-domain where the (only) input to the clustering function is a finite set of points endowed with a between-points distance (or similarity) function, and the output is a partition of that domain.

A *distance function* is a symmetric function $d : X \times X \rightarrow R^+$, such that $d(x, x) = 0$ for all $x \in X$. The data sets that we consider are pairs $(X, d)$, where $X$ is some finite domain set and $d$ is a distance function over $X$. These are the inputs for clustering functions.

A *k-clustering* $C = \{C_1, C_2, \ldots, C_k\}$ of a data set $X$ is a partition of $X$ into $k$ disjoint subsets of $X$ (so, $\bigcup_i C_i = X$). A *clustering* of $X$ is a k-clustering of $X$ for some $1 \leq k \leq |X|$.

For a clustering $C$, let $|C|$ denote the number of clusters in $C$ and $|C_i|$ denote the number of points in a cluster $C_i$. For $x, y \in X$ and a clustering $C$ of $X$, we write $x \sim_C y$ if $x$ and $y$ belong to the same cluster in $C$ and $x \not\sim_C y$, otherwise.

We say that $(X, d)$ and $(X', d')$ are *isomorphic data sets*, denoting it by $(X, d) \sim (X', d')$, if there exists a bijection $\phi : X \rightarrow X'$ so that $d(x, y) = d'(\phi(x), \phi(y))$ for all $x, y \in X$.

We say that two clusterings (or partitions) $C = (c_1, \ldots c_k)$ of some domain $(X, d)$ and $C' = (c'_1, \ldots c'_k)$ of some domain $(X', d')$ are *isomorphic clusterings*, denoted $(C, d) \cong (C', d')$, if there exists a bijection $\phi : X \rightarrow X'$ such that for all $x, y \in X$, $d(x, y) = d'(\phi(x), \phi(y))$ and, on top of that, $x \sim_C y$ if and only if $\phi(x) \sim_{C'} \phi(y)$. Note that this notion depends on both the underlying distance functions and the clusterings.

We consider two definitions of a clustering function.

**Definition 1** (General clustering function). *A general clustering function is a function that takes as input a pair $(X, d)$ and outputs a clustering of the domain $X$.*

The second type are clustering functions that require that the number of clusters be provided as part of the input.

**Definition 2** (k-clustering function). *A k-clustering function is a function that takes as input a pair $(X, d)$ and a parameter $1 \leq k \leq |X|$ and outputs a k-clustering of the domain $X$.*

### 2.1 Properties of Clustering Functions

A key component in our approach are properties of clustering functions that address the input-output behavior of these functions. The properties are formulated for k-clustering functions. However, all the properties, with the exception of locality[1] and refinement-confined, apply also for general clustering functions.

**Isomorphism invariance**: The following invariance property, proposed in [2] under the name "representation independence", seems to be an essential part of our understanding of what clustering is. It requires that the output of a k-clustering function is independent of the labels of the data points. A k-clustering function $F$ is *isomorphism invariant* if whenever $(X, d) \sim (X', d')$, then, for every $k$, $F(X, d, k)$ and $F(X', d', k)$ are isomorphic clusterings.

**Scale invariance**: Scale invariance, proposed by Kleinberg [8], requires that the output of a clustering be invariant to uniform scaling of the data. A k-clustering function $F$ is *scale invariant* if for any data sets $(X, d)$ and $(X, d')$, if there exists a real number $c > 0$ so that for all $x, y \in X$, $d(x, y) = cd'(x, y)$ then for every $1 \leq k \leq |X|$, $F(X, d, k) = F(X, d', k)$.

**Order invariance**: Order invariance, proposed by Jardine and Sibson[6], describes clustering functions that are based on the ordering of pairwise distances. A distance function $d'$ of $X$ is an *order invariant modification* of $d$ over $X$ if for all $x_1, x_2, x_3, x_4 \in X$, $d(x_1, x_2) < d(x_3, x_4)$ if and only if $d'(x_1, x_2) < d'(x_3, x_4)$. A k-clustering function $F$ is *order invariant* if whenever a distance function $d'$ over $X$ is an order invariant modification of $d$, $F(X, d, k) = F(X, d', k)$ for all $k$.

**Locality**: Intuitively, a k-clustering function is local if its behavior on a union of clusters depends only on distances between elements of that union, and is independent of the rest of the domain set. Locality was proposed in [2]. A k-clustering function $F$ is *local* if for any clustering $C$ output by $F$ and every subset of clusters, $C' \subseteq C$, $F(\bigcup C', d, |C'|) = C'$.

In other words, for every domain $(X, d)$ and number of clusters, $k$, if $X'$ is the union of $k'$ clusters in $F(X, d, k)$ for some $k' \leq k$, then, applying $F$ to $(X', d)$ and asking for a $k'$-clustering, will yield the same clusters that we started with.

**Consistency**: Consistency, proposed by Kleinberg [8], aims to formalize the preference for clusters that are dense and well-separated. This property requires that the output of a k-clustering function should remain unchanged after shrinking within-cluster distances and stretching between-cluster distances.

Given a clustering $C$ of some domain $(X, d)$, we say that a distance function $d'$ over $X$, is $(C, d)$-*consistent* if $d'_X(x, y) \leq d_X(x, y)$ whenever $x \sim_C y$, and $d'_X(x, y) \geq d_X(x, y)$ whenever $x \not\sim_C y$. A k-clustering function $F$ is *consistent* if for every $X, d, k$, if $d'$ is $(F(X, d, k), d)$-consistent then $F(X, d, k) = F(X, d', k)$.

While this property may sound desirable and natural, it turns out that many common clustering paradigms fail to satisfy it. In a sense, this property may be viewed as the main weakness of Kleinberg's impossibility result.

The following two properties, proposed in [2], are straightforward relaxations of consistency.

**Inner and Outer consistency**: Outer consistency represents the preference for well separated clusters, by requiring that the output of a k-clustering function not change if clusters are moved away from each other.

A distance function $d'$ over $X$ is $(C, d)$-*outer consistent* if $d'_X(x, y) = d_X(x, y)$ whenever $x \sim_C y$, and $d'_X(x, y) \geq d_X(x, y)$ whenever $x \not\sim_C y$. Outer consistency is defined in the same way consistency, except that $(C, d)$-*consistent* is replaced by $(C, d)$-*outer consistent*.

Inner consistency represents the preference for placing points that are close together within the same cluster, by requiring that the output of a k-clustering function not change if elements of the same cluster are moved closer to each other.

Inner consistency is defined in a similar manner to outer-consistency, except that $d'$ is $(C, d)$-*inner consistent* if $d'_X(x, y) \leq d_X(x, y)$ whenever $x \sim_C y$, and $d'_X(x, y) = d_X(x, y)$ whenever $x \not\sim_C y$.

Clearly, consistency implies both outer-consistency and inner-consistency. Note also that if a function is both inner-consistent and outer-consistent then it is consistent.

**k-Richness**: The k-richness property requires that we be able to obtain any partition of the domain by modifying the distances between elements. This property is based on Kleinberg's [8] richness axiom, requiring that for any sets $X_1, X_2, \ldots, X_k$, there exists a distance function $d$ over $X' = \bigcup_{i=1}^{k} X_i$ so that $F(X', d) = \{X_1, X_2, \ldots, X_k\}$. A k-clustering function $F$ *satisfies k-richness* if for any sets $X_1, X_2, \ldots, X_k$, there exists a distance function $d$ over $X' = \bigcup_{i=1}^{k} X_i$ so that $F(X', d, k) = \{X_1, X_2, \ldots, X_k\}$.

**Outer richness**: Outer richness, a natural variation on the k-richness property, was proposed in [2] under the name "extended richness." (we have renamed it to contrast this property with "inner richness", which we propose in Appendix A). Given $k$ sets, a k-clustering function satisfies outer richness if there exists some way of setting the between-set distances, without modifying distances within the sets, we can get $F$ to output each of these data sets as a cluster. This corresponds to the intuition that any groups of points, regardless of within distances, can be made into separate clusters.

A clustering function $F$ is outer-rich if for every set of domains, $\{(X_1, d_1), \ldots (X_n, d_k)\}$, there exists a distance function $\hat{d}$ over $\bigcup_{i=1}^{n} X_i$ that extends each of the $d_i$'s (for $i \leq k$), such that $F(\bigcup_{i=1}^{k} X_i, \hat{d}, k) = \{X_1, X_2, \ldots, X_k\}$.

**Threshold-richness**: Fundamentally, the goal of clustering is to group points that are close to each other, and to separate points that are far apart. Axioms of clustering need to represent these objectives and no set of axioms of clustering can be complete without integrating such requirements.

| _Function_ | outer consistent | inner consistent | local | refinement-confined | order invariant | k-rich | outer rich | inner rich | threshold rich | scale invariant | iso. invariant |
|---|---|---|---|---|---|---|---|---|---|---|---|
| Single Linkage | ✓ | ✓ | ✓ | ✓ | ✓ | ✓ | ✓ | ✓ | ✓ | ✓ | ✓ |
| Average Linkage | ✓ | X | ✓ | ✓ | X | ✓ | ✓ | ✓ | ✓ | ✓ | ✓ |
| Complete Linkage | ✓ | X | ✓ | ✓ | ✓ | ✓ | ✓ | ✓ | ✓ | ✓ | ✓ |
| $k$-median | ✓ | X | ✓ | X | X | ✓ | ✓ | ✓ | ✓ | ✓ | ✓ |
| $k$-means | ✓ | X | ✓ | X | X | ✓ | ✓ | ✓ | ✓ | ✓ | ✓ |
| Min sum | ✓ | ✓ | ✓ | X | X | ✓ | ✓ | ✓ | ✓ | ✓ | ✓ |
| Ratio cut | X | ✓ | X | X | X | ✓ | ✓ | ✓ | ✓ | ✓ | ✓ |
| Normalized cut | X | X | X | X | X | ✓ | ✓ | ✓ | ✓ | ✓ | ✓ |

Figure 1: A taxonomy of k-clustering functions, illustrating what properties are satisfied by some common k-clustering functions. The results in the $k$-means row apply both when the centers are part of the data set and when the underlying space is Euclidean and the centers are arbitrary points in the space.

Consistency is the only previous property that aims to formalize these requirements. However, consistency has some counterintuitive implications (see Section 3 in [1]), and is not satisfied by many common clustering functions.

A k-clustering function $F$ is *threshold-rich* if for every clustering $C$ of $X$, there exist real numbers $a < b$ so that for every distance function $d$ over $X$ where $d(x, y) \leq a$ for all $x \sim_C y$, and $d(x, y) \geq b$ for all $x \not\sim_C y$, we have that $F(X, d, |C|) = C$.

This property is based on Kleinberg's [8] $\Gamma$-forcing property, and is equivalent to the requirement that for every partition $\Gamma$, there exists $a < b$ so that $(a, b)$ is $\Gamma$-forcing.

**Inner richness**: Complementary to outer richness, inner richness requires that there be a way of setting distances within sets, without modifying distances between the sets, so that $F$ outputs each set as a cluster. This corresponds to the intuition that between-cluster distances cannot eliminate any partition of $X$. A k-clustering function $F$ satisfies *inner richness* if for every data set $(X, d)$ and partition $\{X_1, X_2, \ldots, X_k\}$ of $X$, there exists a $\hat{d}$ where for all $a \in X_i$, $b \in X_j$ for $i \neq j$, $\hat{d}(a, b) = d(a, b)$, and $F(\bigcup_{i=1}^{k} X_i, \hat{d}, k) = \{X_1, X_2, \ldots, X_k\}$.

**Refinement-confined**[2]: The following formalization was proposed in [2]. A clustering $C$ of $X$ is a *refinement* of clustering $C'$ of $X$ if every cluster in $C$ is a subset of some cluster in $C'$, or, equivalently, if every cluster of $C'$ is a union of clusters of $C$. A k-clustering function is *refinement confined* if for every $1 \leq k \leq k' \leq |X|$, $F(X, d, k')$ is a refinement of $F(X, d, k)$.

## 3 Property-Based Classification of Common k-Clustering Functions

In this section we present a taxonomy of common k-clustering functions. The taxonomy is presented in Figure 1 (definitions of the k-clustering functions are in Appendix C in the supplementary material).

The taxonomy in Figure 1 illustrates how clustering algorithms differ from one another. For example, order-invariance and inner-consistency can be used to distinguish among the three common linkage-based algorithms. Min-sum differs from k-means and k-median in that it satisfies inner-consistency. Unlike all the other algorithms discussed, the spectral clustering functions are not local.

The proofs of the claims embedded in the table appear in the supplementary material.

## 3.1 Axioms of clustering

Our taxonomy reveals that some intuitive properties, which may be expected of all k-clustering functions, are not satisfied by some common k-clustering functions. For example, locality is not satisfied by the spectral clustering functions ratio-cut and normalized-cut. Also, most functions fail inner consistency, and therefore do not satisfy consistency, even though the latter was previously proposed as an axiom of k-clustering functions [8].

On the other hand, isomorphism invariance, scale invariance, and all richness properties (in the setting where the number of clusters, $k$, is part of the input), are satisfied by all the clustering functions considered. Isomorphism invariance and scale-invariance make for natural axioms. Threshold richness is the only one that is both satisfied by all k-clustering functions considered and reflects the main objective of clustering: to group points that are close together and to separate points that are far apart.

It is easy to see that threshold richness implies k-richness. It can be shown that when threshold richness is combined with scale invariance, it also implies outer-richness and inner-richness. Therefore, we propose that scale-invariance, isomorphism-invariance, and threshold richness can be used as clustering axioms.

However, we emphasize that these three axioms do not make a complete set of axioms for clustering, since some functions that satisfy all three properties do not make reasonable k-clustering functions; a function that satisfies the two invariance properties can also satisfy threshold richness by behaving reasonably only on particularly well-clusterable data, while having counter-intuitive behavior on other data sets.

# 4 Properties for Randomized k-Clustering Functions

We present a formal setting to study and analyze probabilistic k-clustering functions. A *probabilistic k-clustering function F* takes a data set $(X, d)$ and an integer $1 \leq k \leq |X|$ and outputs $F(X, d, k)$, a probability distribution over k-clusterings of $X$. Let $P(F(X, d, k) = C)$ denote the probability of clustering $C$ in the probability distribution $F(X, d, k)$.

## 4.1 Properties of Probabilistic k-Clustering Functions

We translate properties of different types into the probabilistic setting.

**Invariance properties**: Invariance properties specify when data sets should be clustered in the same way (ex. isomorphism-invariance, scale-invariance, and order-invariance). Such properties are translated into the probabilistic setting by requiring that when data sets $(X, d)$ and $(X', d')$ satisfy some similarity requirements, then $F(X, d, k) = F(X', d', k)$ for all $k$.

**Consistency properties**: Consistency properties impose conditions that should improve the quality of a clustering. Every such property has some notion of a "$(C, d)$-nice" variant that specifies how the underlying distance function can be modified to better flesh out clustering $C$. In the probabilistic setting, such properties require that whenever $d'$ is a $(C, d)$-nice variant, the k-clustering function is at least as likely to output $C$ on $d'$ as on $d$, $P[F(X, d', |C|) = C] \geq P[F(X, d, |C|) = C]$.

**Richness properties**: Richness properties require that any desired clustering can be obtained under certain constraints. In the probabilistic setting, we require that the same occurs with arbitrarily high probability. For example, the following is the probabilistic version of the k-richness property. The other variants of richness are reformulated analogously.

**Definition 3** (k-Richness). *A probabilistic k-clustering function F is* k-rich *if for any k-clustering C of X and any $\epsilon > 0$, there exists a distance function d over X so that $P(F(X, d, k) = C) \geq 1 - \epsilon$.*

**Locality**: We now show how to translate locality into the probabilistic setting. We say that a clustering of $X$ specifies how to cluster a subset $X' \subseteq X$ if every cluster that overlaps with $X'$ is contained within $X'$. Locality requires that a k-clustering function cluster $X'$ in the way specified by the superset $X$.

| Clustering Algorithm | Properties | | Axioms | | | Other | |
|---|---|---|---|---|---|---|---|
| | outer consistent | local | threshold rich | scale invariant | iso. invariant | k-rich | outer rich |
| Optimal $k$-means | ✓ | ✓ | ✓ | ✓ | ✓ | ✓ | ✓ |
| Random Centroids Lloyd | X | X | X | ✓ | ✓ | ✓ | X |
| Furthest Centroids Lloyd | X | X | ✓ | ✓ | ✓ | ✓ | ✓ |

Figure 2: An analysis of the $k$-means clustering function and $k$-means heuristics. The two leftmost properties distinguish the $k$-means clustering function, properties that are satisfied by $k$-means but fail for other reasonable k-clustering functions. The next three are proposed axioms of clustering, and the last two properties follow from the axioms.

In the probabilistic setting, we require that the probability of obtaining a specific clustering of $X' \subseteq X$ is determined by the probability of obtaining that clustering as a subset of $F(X, d, k)$, given that the output of $F$ on $(X, d)$ specifies how to cluster $X'$.

**Definition 4** (Locality (probabilistic)). *A probabilistic k-clustering function $F$ is* local *if for any k-clustering $C'$ of $X'$, $X' \subseteq X$, and $j \geq k$, where $P[\exists C_1, \ldots, C_k \text{ s.t. } \cup_{i=1}^k C_i = X' \mid F(X, d, j) = C] \neq 0$,*

$$P[F(X', d/X', |C'|) = C'] =$$
$$\frac{P[C' \subseteq C \mid F(X, d, j) = C \text{ and } C/X' \text{ is a k-clustering}]}{P[\exists C_1, \ldots, C_k \text{ s.t. } \cup_{i=1}^k C_i = X' \mid F(X, d, j) = C]}.$$

# 5 Properties Distinguishing K-means Heuristics

## 5.1 $k$-means and $k$-means heuristics

One of the most popular clustering algorithms is the Lloyd method, which aims to find clusterings with low $k$-means loss. Indeed, the Lloyd method is sometimes referred to as the "$k$-means algorithm." We maintain a distinction between the $k$-means objective function and heuristics, such as the Lloyd method, which aim to find clusterings with low $k$-means loss. For this section, we assume that the data lie in Euclidean space, as is often the case when the Lloyd method is applied.

**Definition 5** (Lloyd method). *Given a data set $(X, d)$, and a set $S$ of points in $R^n$, the Lloyd algorithm performs the following steps until two consecutive iterations return the same clustering.*

1. *Assign each point in $X$ to its closest element of $S$. That is, find the clustering $C$ of $X$ so that $x \sim_C y$ if and only if $argmin_{c \in S} \|c - x\| = argmin_{c \in S} \|c - y\|$.*

2. *Compute the centers of mass of the clusters. Set $S = \{c_i = \frac{1}{|C_i|} \sum_{x \in C_i} x \mid C_i \in C\}$.*

The Lloyd method is highly sensitive to the choice of initial centers. Perhaps the most common method for initializing the centers for the Lloyd method is to select $k$ random points from the input data set, proposed by Forgy in 1965 [4]. We refer to this initialization method as Random Centroids.

We propose a slight variation on a deterministic initialization method by Katsavounidis, Kuo, and Zhang [7], who propose selecting centers that are far apart. First let $c_1$ and $c_2$ be the two points furthest away from each other. Then, for all $2 \leq k$, let $c_i$ be the point furthest away from its closest existing center. That is, let $c_i$ be the point in $X$ that maximizes $\min_{1 \leq j \leq i-1} d(c_j, c_i)$.

## 5.2 Distinguishing heuristics by properties

An analysis of the $k$-means clustering functions and the two $k$-means heuristics discussed above is shown in Figure 2. The analysis illustrates that the $k$-means function differs significantly from

heuristics that aim to find clusterings with low $k$-means objective loss. The proofs for this analysis were omitted due to a lack of space (they appear in the supplementary material).

There are two properties that are satisfied by the $k$-means clustering function and fail for other reasonable k-clustering functions: outer-consistency and locality. Neither is satisfied by the heuristics.

Note that unlike k-clustering functions that optimize common clustering objective functions, heuristics that aim to find clusterings with low loss for these objective functions do not necessarily make meaningful k-clustering functions. Therefore, such heuristic's failure to satisfy certain properties does not preclude these properties from being axioms of clustering, but rather illustrates a weakness of the heuristic.

It is interesting that the Lloyd method with the Furthest Centroids initialization technique satisfies our proposed axioms of clustering while Lloyd with Random Centroid fails threshold richness. This corresponds to the finding of He et. al. [5] that in practice, Furthest Centroids performs better than Randomized Centroids.

## 6 Impossibility Results

In this final section, we strengthen Kleinberg's famous impossibility result [8], for general clustering functions, yielding a simpler proof of the original result.

Kleinberg impossibility theorem (Theorem 2.1, [8]) was that no general clustering function can simultaneously satisfy scale-invariance, richness, and consistency. Ackerman and Ben-David[1] later showed that consistency has some counter intuitive consequence. In Section 1, we showed that many natural clustering functions fail inner-consistency[3], which implies that there are many general clustering functions that fail consistency.

On the other hand, many natural algorithms satisfy outer-consistency. We strengthen Kleinberg's impossibility result by relaxing consistency to outer-consistency.

**Theorem 1.** *No general clustering function can simultaneously satisfy outer-consistency, scale-invariance, and richness.*

*Proof.* Let $F$ be any general clustering function that satisfies outer-consistency, scale-invariance and richness.

Let $X$ be some domain set with three or more elements. By richness, there exist distance functions $d_1$ and $d_2$ such that $F(X, d_1) = \{X\}$ (every domain point is a cluster on its own) and $F(X, d_2)$ is some different clustering, $C = \{C_1, \ldots C_k\}$ of $X$.

Let $r = \max\{d_1(x, y) : x, y \in X\}$ and let $c$ be such that for every $x \neq y$, $cd_2(x, y) \geq r$. Define $\hat{d}(x, y) = c \cdot d_2(x, y)$, for every $x, y \in X$. Note that $\hat{d}(x, y) \geq d_1(x, y)$ for all $x, y \in X$. By outer-consistency, $F(X, \hat{d}) = F(X, d_1)$. However, by scale-invariance $F(X, \hat{d}) = F(X, d_2)$. This is a contradiction since $F(X, d_1)$ and $F(X, d_2)$ are different clusterings. $\square$

A similar result can be obtained, using a similar proof, with inner-consistency replacing outer consistency. Namely,

**Lemma 1.** *No general clustering function can simultaneously satisfy inner-consistency, scale-invariance, and richness.*

Since consistency implies both outer-consistency and inner-consistency, Kleinberg's original result follows from any one of Theorem 1 or Lemma 1.

Kleinberg's impossibility result illustrates property trade-offs for general clustering functions. The good news is that these results do not apply when the number of clusters is part of the input, as is illustrated in our taxonomy; single linkage satisfies scale-invariance, consistency and richness.

## Footnotes

[1] Locality can also be reformulated for general clustering functions, however, we do not discuss this in this work.

[2]In [2], this property was called "hierarchical clustering".

[3]Note that a clustering function and it's corresponding general clustering function satisfy the same set of consistency properties.

# References

[1] M. Ackerman and S. Ben-David. Measures of Clustering Quality: A Working Set of Axioms for Clustering. NIPS, 2008.

[2] M. Ackerman, S. Ben-David, and D. Loker. Characterization of Linkage-based Clustering. COLT, 2010.

[3] R. Bosagh Zadeh and S. Ben-David. "A Uniqueness Theorem for Clustering." The 25th Annual Conference on Uncertainty in Artificial Intelligence UAI, 2009.

[4] E. Forgy. Cluster analysis of multivariate data: efficiency vs. interpretability of classifications. In WNAR meetings, Univ of Calif Riverside, number 768, 1965.

[5] He, J., Lan, M., Tan, C.-L., Sung, S. -Y., and Low, H.-B. (2004). Initialization of cluster refinement algorithms: A review and comparative study. In Proc. IEEE Int. Joint Conf. Neural Networks (pp. 297?-302).

[6] N. Jardine, R. Sibson, Mathematical Taxonomy Wiley, 1971.

[7] I. Katsavounidis, C.-C. J. Kuo, and Z. Zhang. A new initialization technique for generalized Lloyd iteration. IEEE Signal Processing Letters, 1(10):144-146, 1994.

[8] Jon Kleinberg. "An Impossibility Theorem for Clustering." Advances in Neural Information Processing Systems (NIPS) 15, 2002.

[9] U. von Luxburg. A Tutorial on Spectral Clustering. Statistics and Computing 17(4): 395-416, 2007

[10] L. Vendramin, R.J.G.B. Campello, and E.R. Hruschka. "On the comparison of relative clustering validity criteria." Sparks, 2009.

